# A MODEL FOR RESOLUTION ENHANCEMENT (HYPERACUITY) IN SENSORY REPRESENTATION

Jun Zhang and John P. Miller

*Neurobiology Group, University of California,*
*Berkeley, California 94720, U.S.A.*

## ABSTRACT

Heiligenberg (1987) recently proposed a model to explain how sensory maps could enhance resolution through orderly arrangement of broadly tuned receptors. We have extended this model to the general case of polynomial weighting schemes and proved that the response function is also a polynomial of the same order. We further demonstrated that the Hermitian polynomials are eigenfunctions of the system. Finally we suggested a biologically plausible mechanism for sensory representation of external stimuli with resolution far exceeding the inter-receptor separation.

## 1  INTRODUCTION

In sensory systems, the stimulus continuum is sampled at discrete points by receptors of finite tuning width $d$ and inter-receptor spacing $a$. In order to code both stimulus locus and stimulus intensity with a single output, the sampling of individual receptors must be overlapping (i.e. $a < d$). This discrete and overlapped sampling of the stimulus continuum poses a question of how then the system could reconstruct the sensory stimuli with

a resolution exceeding that is specified by inter-receptor spacing. This is known as the hyperacuity problem (Westheimer,1975).

Heiligenberg (1987) proposed a model in which the array of receptors (with Gaussian-shaped tuning curves) were distributed uniformly along the entire range of stimulus variable $x$. They contribute excitation to a higher order interneuron, with the synaptic weight of each receptor's input set proportional to its rank index $k$ in the receptor array. Numerical simulation and subsequent mathematical analysis (Baldi and Heiligenberg, 1988) demonstrated that, so long as $a \ll d$, the response function $f(x)$ of the higher order neuron was monotone increasing and surprisingly linear. The smoothness of this function offers a partial explanation of the general phenomena of hyperacuity (see Baldi and Heiligenberg in this volumn). Here we consider various extensions of this model. Only the main results shall be stated below; their proof is presented elsewhere (Zhang and Miller, in preparation).

## 2   POLYNOMIAL WEIGHTING FUNCTIONS

First, the model can be extended to incorporate other different weighting schemes. The weighting function $w(k)$ specifies the strength of the excitation from the $k$-th receptor onto the higher order interneuron and therefore determines the shape of its response $f(x)$. In Heiligenberg's original model, the linear weighting scheme $\omega(k) = k$ is used. A natural extension would then be the polynomial weighting schemes. Indeed, we proved that, for sufficiently large $d$,

a) If $\omega(k) = k^{2m}$ , then:

$$f(x) = a_0 + a_2 x^2 + \ldots + a_{2m} x^{2m}$$

If $\omega(k) = k^{2m+1}$, then:

$$f(x) = a_1 x + a_3 x^3 + \ldots + a_{2m+1} x^{2m+1}$$

where $m = 0, 1, 2, \ldots$, and $a_i$ are real constants.

Note that for $\omega(k) = k^p$, $f(x)$ has parity $(-1)^p$, that is, it is an odd function for odd interger $p$ and even function for even interger $p$. The case of $p = 1$ reduces to the linear weighting scheme in Heiligenberg's original model.

b)  If $\omega(k) = c_0 + c_1 k + c_2 k^2 + \ldots + c_p k^p$, then:

$$f(x) = a_0 + a_1 x + a_2 x^2 + \ldots + a_p x^p$$

Note that this is a direct result of a), because $f(x)$ is linearly dependent on $\omega(k)$. The coefficients $c_i$ and $a_i$ are usually different for the two polynomials. One would naturally ask: what kind of polynomial weighting function then would yield an identical polynomial response function? This leads to the important conclusion:

c)  If $\omega(k) = H_p(k)$ is an Hermitian polynomial, then $f(x) = H_p(x)$, the same Hermitian polynomial.

The Hermitian polynomial $H_p(t)$ is a well-studied function in mathematics. It is defined as:

$$H_p(t) = (-1)^p e^{t^2} \frac{d^p}{dt^p} e^{-t^2}$$

For reference purpose, the first four polynomials are given here:

$$
\begin{aligned}
H_0(t) &= 1; \\
H_1(t) &= 2t; \\
H_2(t) &= 4t^2 - 2; \\
H_3(t) &= 8t^3 - 12t;
\end{aligned}
$$

The conclusion of c) tells us that Hermitian polynomials are unique in the sense that they serve as eigenfunctions of the system.

## 3    REPRESENTATION OF SENSORY STIMULUS

Heiligenberg's model deals with the general problem of two-point resolution, i.e. how sensory system can resolve two nearby point stimuli with a resolution exceeding inter-receptor spacing. Here we go one step further to ask ourselves how a generalized sensory stimulus $g(x)$ is encoded and represented beyond the receptor level with a resolution exceeding the inter-receptor spacing. We'll show that if, instead of a single higher order interneuron, we have a group or layer of interneurons, each connected to the array of sensory receptors using some different but appropriately chosen weighting schemes $w_n(k)$, then the representation of the sensory stimulus by this interneuron group (in terms of $f_n$ , each interneuron's response) is uniquely determined *with enhanced resolution* (see figure below).

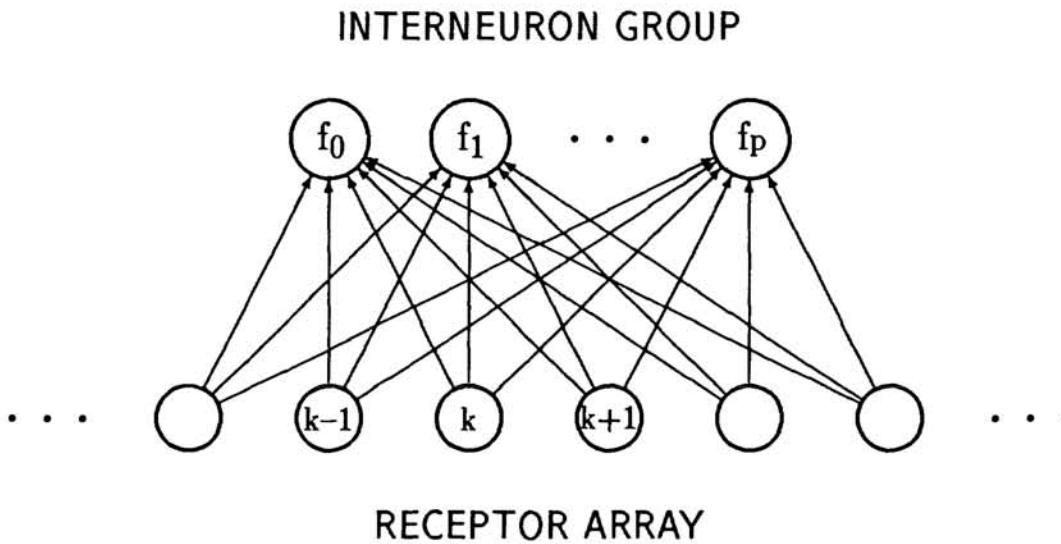

INTERNEURON GROUP

RECEPTOR ARRAY

Suppose that 1) each interneuron in this group receives input from the receptor array, its weighting characterized by a Hermitian polynomial $H_p(k)$; and that 2) the order $p$ of the Hermitian polynomial is different for each interneuron. We know from mathematics that any stimulus function $g(x)$ satisfying certain boundary conditions can be decomposed in the following way:

$$g(x) = \sum_{n=0}^{\infty} c_n H_n(x) e^{-x^2}$$

The decomposition is unique in the sense that $c_n$ completely determines $g(x)$. Here we have proved that the response $f_p$ of the $p$-th interneuron (adopting $H_p(k)$ as weighting scheme) is proportional to $c_p$:

$$f_p \propto c_p$$

This implies that $g(x)$ can be uniquely represented by the response of this set of interneurons $\{\, f_p \,\}$. Note that the precision of representation at this higher stage is limited not by the receptor separation, but by the number of neurons available in this interneuron group.

## 4   EDGE EFFECTS

Since the array of receptors must actually be finite in extent, simple weighting schemes may result in edge-effects which severely degrade stimulus resolution near the array boundaries. For instance, the linear model investigated by Heiligenberg and Baldi will have regions of degeneracy where two nearby point stimuli, if located near the boundary defined by receptor array coverage, may yield the same response. We argue that this region of degeneracy can be eliminated or reduced in the following situations:

1) If $\omega(k)$ approaches zero as $k$ goes to infinity, then the receptor array

can still be treated as having infinite extent since the contributions by the large index receptors are negligibly small. We proved, using Fourier analysis, that this kind of vanishing-at-infinity weighting scheme could also achieve resolution enhancement provided that the tuning width of the receptor is sufficiently larger than the inter-receptor spacing and meanwhile sufficiently smaller than the effective width of the entire weighting function.

2) If the receptor array "wraps around" into a circular configuration, then it can again be treated as infinite (but periodic) along the angular dimension. This is exactly the case in the wind-sensitive cricket cercal sensory system (Jacobs et al,1986; Jacobs and Miller,1988) where the population of directional selective mechano-receptors covers the entire range of 360 degrees.

## 5  CONCLUSION

Heiligenberg's model, which employs an array of orderly arranged and broadly tuned receptors to enhance the two-point resolution, can be extended in a number of ways. We first proved the general result that the model works for any polynomial weighting scheme. We further demonstrated that Hermitian polynomial is the eigenfunction of this system. This leads to the new concept of stimulus representation, i.e. a group of higher-order interneurons can encode any generalized sensory stimulus with enhanced resolution if they adopt appropriately chosen weighting schemes. Finally we discussed possible ways of eliminating or reducing the "edge-effects".

## ACKNOWLEDGMENTS

This work was supported by NIH grant # R01-NS26117.

## REFERENCES

Baldi, P. and W. Heiligenberg (1988) How sensory maps could enhance resolution through ordered arrangements of broadly tuned receivers. *Biol. Cybern.* 59: 314-318.

Heiligenberg, W. (1987) Central processing of the sensory information in electric fish. *J. Comp. Physiol. A* 161: 621-631.

Jacobs, G.A. and J.P. Miller (1988) Analysis of synaptic integration using the laser photo-inactivation technique. *Experientia* 44: 361- 462.

Jacobs, G.A., Miller, J.P. and R.K. Murphey (1986) Cellular mechanisms underlying directional sensitivity of an identified sensory interneuron. *J.Neurosci.* 6: 2298-2311.

Westheimer, G. (1975) Visual acuity and hyperacuity. *Invest. Ophthalmol. Vis.* 14: 570-572.